# Mismatch String Kernels for SVM Protein Classification

**Christina Leslie**
Department of Computer Science
Columbia University
cleslie@cs.columbia.edu

**Eleazar Eskin**
Department of Computer Science
Columbia University
eeskin@cs.columbia.edu

**Jason Weston**
Max-Planck Institute
Tuebingen, Germany
weston@tuebingen.mpg.de

**William Stafford Noble***
Department of Genome Sciences
University of Washington
noble@gs.washington.edu

## Abstract

We introduce a class of string kernels, called mismatch kernels, for use with support vector machines (SVMs) in a discriminative approach to the protein classification problem. These kernels measure sequence similarity based on shared occurrences of $k$-length subsequences, counted with up to $m$ mismatches, and do not rely on any generative model for the positive training sequences. We compute the kernels efficiently using a mismatch tree data structure and report experiments on a benchmark SCOP dataset, where we show that the mismatch kernel used with an SVM classifier performs as well as the Fisher kernel, the most successful method for remote homology detection, while achieving considerable computational savings.

## 1   Introduction

A fundamental problem in computational biology is the classification of proteins into functional and structural classes based on *homology* (evolutionary similarity) of protein sequence data. Known methods for protein classification and homology detection include pairwise sequence alignment [1, 2, 3], profiles for protein families [4], consensus patterns using motifs [5, 6] and profile hidden Markov models [7, 8, 9]. We are most interested in discriminative methods, where protein sequences are seen as a set of labeled examples — positive if they are in the protein family or superfamily and negative otherwise — and we train a classifier to distinguish between the two classes. We focus on the more difficult problem of *remote homology detection*, where we want our classifier to detect (as positives) test sequences that are only remotely related to the positive training sequences.

One of the most successful discriminative techniques for protein classification – and the best performing method for remote homology detection – is the Fisher-SVM [10, 11] approach of Jaakkola *et al.* In this method, one first builds a profile hidden Markov model

(HMM) for the positive training sequences, defining a log likelihood function $\log P(x|\theta)$ for any protein sequence $x$. If $\theta_o$ is the maximum likelihood estimate for the model parameters, then the gradient vector $\nabla_\theta \log P(x|\theta)\big|_{\theta=\theta_o}$ assigns to each (positive or negative) training sequence $x$ an explicit vector of features called Fisher scores. This feature mapping defines a kernel function, called the *Fisher kernel*, that can then be used to train a support vector machine (SVM) [12, 13] classifier. One of the strengths of the Fisher-SVM approach is that it combines the rich biological information encoded in a hidden Markov model with the discriminative power of the SVM algorithm. However, one generally needs a lot of data or sophisticated priors to train the hidden Markov model, and because calculating the Fisher scores requires computing forward and backward probabilities from the Baum-Welch algorithm (quadratic in sequence length for profile HMMs), in practice it is very expensive to compute the kernel matrix.

In this paper, we present a new string kernel, called the mismatch kernel, for use with an SVM for remote homology detection. The $(k, m)$-mismatch kernel is based on a feature map to a vector space indexed by all possible subsequences of amino acids of a fixed length $k$; each instance of a fixed $k$-length subsequence in an input sequence contributes to all feature coordinates differing from it by at most $m$ mismatches. Thus, the mismatch kernel adds the biologically important idea of mismatching to the computationally simpler spectrum kernel presented in [14]. In the current work, we also describe how to compute the new kernel efficiently using a mismatch tree data structure; for values of $(k, m)$ useful in this application, the kernel is fast enough to use on real datasets and is considerably less expensive than the Fisher kernel. We report results from a benchmark dataset on the SCOP database [15] assembled by Jaakkola *et al.* [10] and show that the mismatch kernel used with an SVM classifier achieves performance equal to the Fisher-SVM method while outperforming all other methods tested. Finally, we note that the mismatch kernel does not depend on any generative model and could potentially be used in other sequence-based classification problems.

## 2   Spectrum and Mismatch String Kernels

The basis for our approach to protein classification is to represent protein sequences as vectors in a high-dimensional feature space via a string-based *feature map*. We then train a support vector machine (SVM), a large-margin linear classifier, on the feature vectors representing our training sequences. Since SVMs are a kernel-based learning algorithm, we do not calculate the feature vectors explicitly but instead compute their pairwise inner products using a *mismatch string kernel*, which we define in this section.

### 2.1   Feature Maps for Strings

The $(k, m)$-mismatch kernel is based on a feature map from the space of all finite sequences from an alphabet $\mathcal{A}$ of size $|\mathcal{A}| = l$ to the $l^k$-dimensional vector space indexed by the set of $k$-length subsequences ("$k$-mers") from $\mathcal{A}$. (For protein sequences, $\mathcal{A}$ is the alphabet of amino acids, $l = 20$.) For a fixed $k$-mer $\alpha = a_1 a_2 \ldots a_k$, with each $a_i$ a character in $\mathcal{A}$, the $(k, m)$-neighborhood generated by $\alpha$ is the set of all $k$-length sequences $\beta$ from $\mathcal{A}$ that differ from $\alpha$ by at most $m$ mismatches. We denote this set by $N_{(k,m)}(\alpha)$.

We define our feature map $\Phi_{(k,m)}$ as follows: if $\alpha$ is a $k$-mer, then

$$\Phi_{(k,m)}(\alpha) = (\phi_\beta(\alpha))_{\beta \in \mathcal{A}^k} \tag{1}$$

where $\phi_\beta(\alpha) = 1$ if $\beta$ belongs to $N_{(k,m)}(\alpha)$, and $\phi_\beta(\alpha) = 0$ otherwise. Thus, a $k$-mer contributes weight to all the coordinates in its mismatch neighborhood.

For a sequence $x$ of any length, we extend the map additively by summing the feature

vectors for all the $k$-mers in $x$:

$$\Phi_{(k,m)}(x) = \sum_{k\text{-mers } \alpha \text{ in } x} \Phi_{(k,m)}(\alpha)$$

Note that the $\beta$-coordinate of $\Phi_{(k,m)}(x)$ is just a count of all instances of the $k$-mer $\beta$ occurring with up to $m$ mismatches in $x$. The $(k,m)$-mismatch kernel $K_{(k,m)}$ is the inner product in feature space of feature vectors:

$$K_{(k,m)}(x,y) = \langle \Phi_{(k,m)}(x), \Phi_{(k,m)}(y)\rangle.$$

For $m = 0$, we retrieve the $k$-spectrum kernel defined in [14].

## 2.2 Fisher Scores and the Spectrum Kernel

While we define the spectrum and mismatch feature maps without any reference to a generative model for the positive class of sequences, there is some similarity between the $k$-spectrum feature map and the Fisher scores associated to an order $k - 1$ Markov chain model. More precisely, suppose the generative model for the positive training sequences is given by

$$P(x|\theta) = P(x_1 \ldots x_{k-1}|\theta)P(x_k|x_1 \ldots x_{k-1}, \theta) \ldots P(x_n|x_{n-k+1} \ldots x_{n-1}, \theta)$$

for a string $x = x_1 x_2 \ldots x_n$, with parameters

$$P(x_j = t | x_{j-k+1} \ldots x_{j-1} = s_1 \ldots s_{k-1}, \theta) = \theta^{t|s_1 \ldots s_{k-1}}$$

for characters $t, s_1, \ldots, s_{k-1}$ in alphabet $\mathcal{A}$. Denote by $\theta_o$ the maximum likelihood estimate for $\theta$ on the positive training set. To calculate the Fisher scores for this model, we follow [10] and define independent variables $\theta^{t, s_1 \ldots s_{k-1}} = \frac{\theta^{t|s_1 \ldots s_{k-1}}}{\sum_{t'} \theta^{t'|s_1 \ldots s_{k-1}}}$ satisfying $\theta_o^{t, s_1 \ldots s_{k-1}} = \theta_o^{t|s_1 \ldots s_{k-1}}$, $\sum_{t'} \theta_o^{t', s_1 \ldots s_{k-1}} = 1$. Then the Fisher scores are given by

$$\frac{\partial}{\partial \theta^{t, s_1 \ldots s_{k-1}}} \log P(x|\theta)\bigg|_{\theta = \theta_o} = n_{t|s_1 \ldots s_{k-1}} \left( \frac{1 - \theta_o^{t, s_1 \ldots s_{k-1}}}{\theta_o^{t, s_1 \ldots s_{k-1}}} \right) - \sum_{\tilde{t} \neq t} n_{\tilde{t}|s_1 \ldots s_{k-1}}$$

$$= \frac{n_{t|s_1 \ldots s_{k-1}}}{\theta_o^{t|s_1 \ldots s_{k-1}}} - n_{s_1 \ldots s_{k-1}},$$

where $n_{t|s_1 \ldots s_{k-1}}$ is the number of instances of the $k$-mer $s_1 \ldots s_{k-1}t$ in $x$, and $n_{s_1 \ldots s_{k-1}}$ is the number of instances of the $(k-1)$-mer $s_1 \ldots s_{k-1}$. Thus the Fisher score captures the degree to which the $k$-mer $s_1 \ldots s_{k-1}t$ is over- or under-represented relative to the positive model. For the $k$-spectrum kernel, the corresponding feature coordinate looks similar but simply uses the unweighted count: $\phi_{s_1 \ldots s_{k-1}t}(x) = n_{t|s_1 \ldots s_{k-1}}$.

# 3 Efficient Computation of the Mismatch Kernel

Unlike the Fisher vectors used in [10], our feature vectors are sparse vectors in a very high dimensional feature space. Thus, instead of calculating and storing the feature vectors, we directly and efficiently compute the kernel matrix for use with an SVM classifier.

## 3.1 Mismatch Tree Data Structure

We use a *mismatch tree* data structure (similar to a trie or suffix tree [16, 17]) to represent the feature space (the set of all $k$-mers) and perform a lexical traversal of all $k$-mers occurring in the sample dataset match with up to $m$ of mismatches; the entire kernel matrix

$K(x_i, x_j)$, $i, j = 1 \ldots M$ for the sample of $M$ sequences is computed in one traversal of the tree.

A $(k, m)$-mismatch tree is a rooted tree of depth $k$ where each internal node has $|\mathcal{A}| = l$ branches and each branch is labeled with a symbol from $\mathcal{A}$. A leaf node represents a fixed $k$-mer in our feature space – obtained by concatenating the branch symbols along the path from root to leaf – and an internal node represents the prefix for those $k$-mer features which are its descendants in the tree. We use a depth-first search of this tree to store, at each node that we visit, a set of pointers to all instances of the current prefix pattern that occur with mismatches in the sample data. Thus at each node of depth $d$, we maintain pointers to all substrings from the sample data set whose $d$-length prefixes are within $m$ mismatches from the $d$-length prefix represented by the path down from the root. Note that the set of valid substrings at a node is a subset of the set of valid substrings of its parent. When we encounter a node with an empty list of pointers (no valid occurrences of the current prefix), we do not need to search below it in the tree. When we reach a leaf node, we sum the contributions of all instances occurring in each source sequence to obtain feature values corresponding to the current $k$-mer, and we update the kernel matrix entry $K(x_a, x_b)$ for each pair of source sequences $x_a$ and $x_b$ having non-zero feature values.

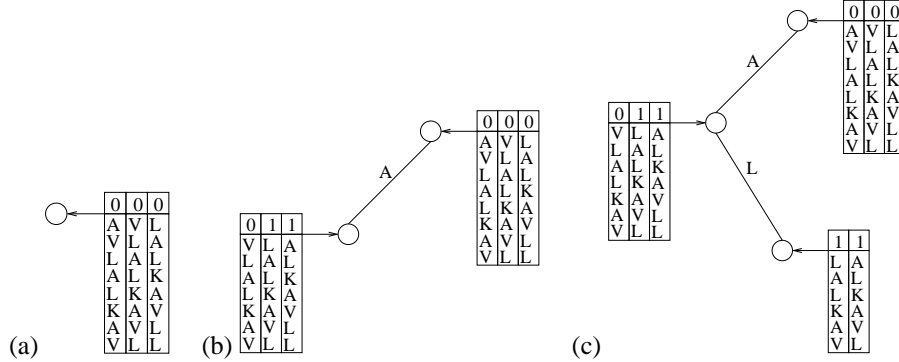

Figure 1: An $(8, 1)$-mismatch tree for a sequence AVLALKAVLL, showing valid instances at each node down a path: (a) at the root node; (b) after expanding the path $A$; and (c) after expanding the path $AL$. The number of mismatches for each instance is also indicated.

## 3.2 Efficiency of the Kernel Computation

Since we compute the kernel in one depth-first traversal, we do not actually need to store the entire mismatch tree but instead compute the kernel using a recursive function, which makes more efficient use of memory and allows kernel computations for large datasets.

The number of $k$-mers within $m$ mismatches of any given fixed $k$-mer is $p(k, m, l) = \sum_{i=0}^{m} \binom{k}{i} (l-1)^i = O(k^m l^m)$. Thus the effective number of $k$-mer instances that we need to traverse grows as $O(Nk^m l^m)$, where $N$ is the total length of the sample data. At a leaf node, if exactly $c$ input sequences contain valid instances of the current $k$-mer, one performs $c^2$ updates to the kernel matrix. For $M$ sequences each of length $n$ (total length $N = nM$), the worst case for the kernel computation occurs when the $M$ feature vectors are all equal and have the maximal number of non-zero entries, giving worst case overall running time $O(M^2 np(k, m, l)) = O(M^2 nk^m l^m)$. For the application we discuss here, small values of $m$ are most useful, and the kernel calculations are quite inexpensive.

When mismatch kernels are used in combination with SVMs, the learned classifier $f(x) =$

$\sum_{i=1}^{r} y_i \alpha_i \langle \Phi_{(k,m)}(x_i), \Phi_{(k,m)}(x) \rangle + b$ (where $x_i$ are the training sequences that map to support vectors, $y_i$ are labels, and $\alpha_i$ are weights) can be implemented by pre-computing and storing per $k$-mer scores. Then the prediction $f(x)$ can be calculated in linear time by look-up of $k$-mer scores. In practice, one usually wants to use a normalized feature map, so one would also need to compute the norm of the vector $\Phi_{(k,m)}(x)$, with complexity $O(nk^m l^m)$ for a sequence of length $n$. Simple $O(1)$ normalization schemes, like dividing by sequence length, can also be used.

## 4   Experiments: Remote Protein Homology Detection

We test the mismatch kernel with an SVM classifier on the SCOP [15] (version 1.37) datasets designed by Jaakkola *et al.* [10] for the remote homology detection problem. In these experiments, remote homology is simulated by holding out all members of a target SCOP family from a given superfamily. Positive training examples are chosen from the remaining families in the same superfamily, and negative test and training examples are chosen from disjoint sets of folds outside the target family's fold. The held-out family members serve as positive test examples. In order to train HMMs, Jaakkola *et al.* used the SAM-T98 algorithm to pull in domain homologs from the non-redundant protein database and added these sequences as positive examples in the experiments. Details of the datasets are available at www.soe.ucsc.edu/research/compbio/discriminative.

Because the test sets are designed for remote homology detection, we use small values of $k$. We tested $(k,m) = (5,1)$ and $(6,1)$, where we normalized the kernel via $K_{(k,m)}^{\text{Norm}}(x,y) = \dfrac{K_{(k,m)}(x,y)}{\sqrt{K_{(k,m)}(x,x)} \sqrt{K_{(k,m)}(y,y)}}$. We found that $(k,m) = (5,1)$ gave slightly better performance, though results were similar for the two choices. (Data for $(k,m) = (6,1)$ not shown.) We use a publicly available SVM implementation (www.cs.columbia.edu/compbio/svm) of the soft margin optimization algorithm described in [10]. For comparison, we include results from three other methods. These include the original experimental results from Jaakkola *et al.* for two methods: the SAM-T98 iterative HMM, and the Fisher-SVM method. We also test PSI-BLAST [3], an alignment-based method widely used in the biological community, on the same data using the methodology described in [14].

Figure 2 illustrates the mismatch-SVM method's performance relative to three existing homology detection methods as measured by ROC scores. The figure includes results for all 33 SCOP families, and each series corresponds to one homology detection method. Qualitatively, the curves for Fisher-SVM and mismatch-SVM are quite similar. When we compare the overall performance of two methods using a two-tailed signed rank test [18, 19] based on ROC scores over the 33 families with a $p$-value threshold of 0.05 and including a Bonferroni adjustment to account for multiple comparisons, we find only the following significant differences: Fisher-SVM and mismatch-SVM perform better than SAM-T98 (with p-values 1.3e-02 and 2.7e-02, respectively); and these three methods all perform significantly better than PSI-BLAST in this experiment.

Figure 3 shows a family-by-family comparison of performance of the $(5,1)$-mismatch-SVM and Fisher-SVM using ROC scores in plot (A) and ROC-50 scores in plot (B). [1] In both plots, the points fall approximately evenly above and below the diagonal, indicating little difference in performance between the two methods. Figure 4 shows the improvement provided by including mismatches in the SVM kernel. The figures plot ROC scores (plot

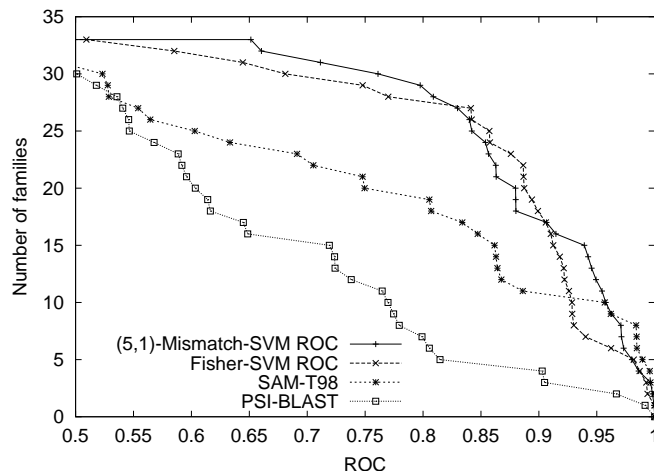

Figure 2: **Comparison of four homology detection methods.** The graph plots the total number of families for which a given method exceeds an ROC score threshold.

(A)) and ROC-50 scores (plot (B)) for two string kernel SVM methods: using $k = 5, m = 1$ mismatch kernel, and using $k = 3$ (no mismatch) spectrum kernel, the best-performing choice with $m = 0$. Almost all of the families perform better with mismatching than without, showing that mismatching gives significantly better generalization performance.

## 5   Discussion

We have presented a class of string kernels that measure sequence similarity without requiring alignment or depending upon a generative model, and we have given an efficient method for computing these kernels. For the remote homology detection problem, our discriminative approach — combining support vector machines with the mismatch kernel — performs as well in the SCOP experiments as the most successful known method.

A practical protein classification system would involve fast multi-class prediction – potentially involving thousands of binary classifiers – on massive test sets. In such applications, computational efficiency of the kernel function becomes an important issue. Chris Watkins [20] and David Haussler [21] have recently defined a set of kernel functions over strings, and one of these string kernels has been implemented for a text classification problem [22]. However, the cost of computing each kernel entry is $O(n^2)$ in the length of the input sequences. Similarly, the Fisher kernel of Jaakkola *et al.* requires quadratic-time computation for each Fisher vector calculated. The $(k, m)$-mismatch kernel is relatively inexpensive to compute for values of $m$ that are practical in applications, allows computation of multiple kernel values in one pass, and significantly improves performance over the previously presented (mismatch-free) spectrum kernel.

Many family-based remote homogy detection algorithms incorporate a method for selecting probable domain homologs from unannotated protein sequence databases for additional training data. In these experiments, we used the domain homologs that were identified by SAM-T98 (an iterative HMM-based algorithm) as part of the Fisher-SVM method and included in the datasets; these homologs may be more useful to the Fisher kernel than to the mismatch kernel. We plan to extend our method by investigating semi-supervised techniques for selecting unannotated sequences for use with the mismatch-SVM.

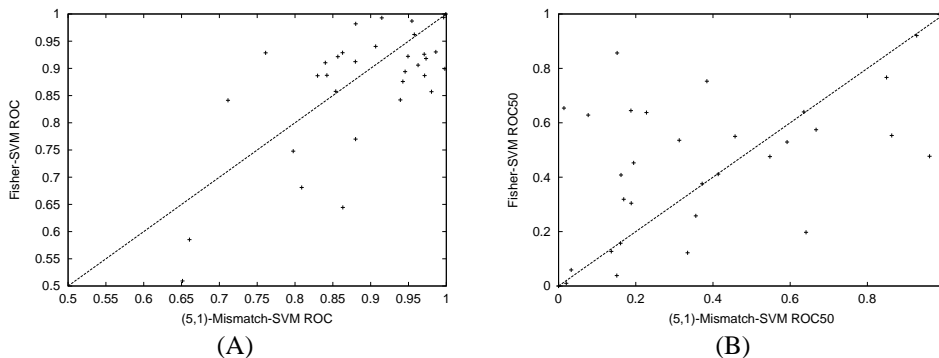

Figure 3: **Family-by-family comparison of** $(5, 1)$**-mismatch-SVM with Fisher-SVM.** The coordinates of each point in the plot are the ROC scores (plot (A)) or ROC-50 scores (plot (B)) for one SCOP family, obtained using the mismatch-SVM with $k = 5$, $m = 1$ (x-axis) and Fisher-SVM (y-axis). The dotted line is $y = x$.

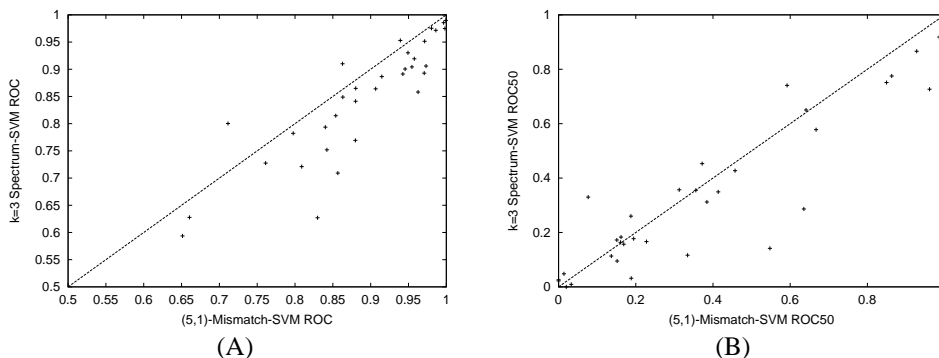

Figure 4: **Family-by-family comparison of** $(5, 1)$**-mismatch-SVM with spectrum-SVM.** The coordinates of each point in the plot are the ROC scores (plot (A)) or ROC-50 scores (plot (B)) for one SCOP family, obtained using the mismatch-SVM with $k = 5$, $m = 1$ (x-axis) and spectrum-SVM with $k = 3$ (y-axis). The dotted line is $y = x$.

Many interesting variations on the mismatch kernel can be explored using the framework presented here. For example, explicit $k$-mer feature selection can be implemented during calculation of the kernel matrix, based on a criterion enforced at each leaf or internal node. Potentially, a good feature selection criterion could improve performance in certain applications while decreasing kernel computation time. In biological applications, it is also natural to consider weighting each $k$-mer instance contribution to a feature coordinate by evolutionary substitution probabilities. Finally, one could use linear combinations of kernels $K_{(k_i, m_i)}$ to capture similarity of different length $k$-mers. We believe that further experimentation with mismatch string kernels could be fruitful for remote protein homology detection and other biological sequence classification problems.

**Acknowledgments**

CL is partially supported by NIH grant LM07276-02. WSN is supported by NSF grants DBI-0078523 and ISI-0093302. We thank Nir Friedman for pointing out the connection with Fisher scores for Markov chain models.

## Footnotes

*Formerly William Noble Grundy: see http://www.cs.columbia.edu/~noble/name-change.html

[1] The ROC-50 score is the area under the graph of the number of true positives as a function of false positives, up to the first 50 false positives, scaled so that both axes range from 0 to 1. This score is sometimes preferred in the computational biology community, motivated by the idea that a biologist might be willing to sift through about 50 false positives.

# References

[1] M. S. Waterman, J. Joyce, and M. Eggert. *Computer alignment of sequences*, chapter Phylogenetic Analysis of DNA Sequences. Oxford, 1991.

[2] S. F. Altschul, W. Gish, W. Miller, E. W. Myers, and D. J. Lipman. A basic local alignment search tool. *Journal of Molecular Biology*, 215:403–410, 1990.

[3] S. F. Altschul, T. L. Madden, A. A. Schaffer, J. Zhang, Z. Zhang, W. Miller, and D. J. Lipman. Gapped BLAST and PSI-BLAST: A new generation of protein database search programs. *Nucleic Acids Research*, 25:3389–3402, 1997.

[4] Michael Gribskov, Andrew D. McLachlan, and David Eisenberg. Profile analysis: Detection of distantly related proteins. *PNAS*, pages 4355–4358, 1987.

[5] A. Bairoch. The PROSITE database, its status in 1995. *Nucleic Acids Research*, 24:189–196, 1995.

[6] T. K. Attwood, M. E. Beck, D. R. Flower, P. Scordis, and J. N Selley. The PRINTS protein fingerprint database in its fifth year. *Nucleic Acids Research*, 26(1):304–308, 1998.

[7] A. Krogh, M. Brown, I. Mian, K. Sjolander, and D. Haussler. Hidden markov models in computational biology: Applications to protein modeling. *Journal of Molecular Biology*, 235:1501–1531, 1994.

[8] S. R. Eddy. Multiple alignment using hidden markov models. In *Proceedings of the Third International Conference on Intelligent Systems for Molecular Biology*, pages 114–120. AAAI Press, 1995.

[9] P. Baldi, Y. Chauvin, T. Hunkapiller, and M. A. McClure. Hidden markov models of biological primary sequence information. *PNAS*, 91(3):1059–1063, 1994.

[10] T. Jaakkola, M. Diekhans, and D. Haussler. A discriminative framework for detecting remote protein homologies. *Journal of Computational Biology*, 2000.

[11] T. Jaakkola, M. Diekhans, and D. Haussler. Using the fisher kernel method to detect remote protein homologies. In *Proceedings of the Seventh International Conference on Intelligent Systems for Molecular Biology*, pages 149–158. AAAI Press, 1999.

[12] V. N. Vapnik. *Statistical Learning Theory*. Springer, 1998.

[13] N. Cristianini and J. Shawe-Taylor. *An Introduction to Support Vector Machines.* Cambridge, 2000.

[14] C. Leslie, E. Eskin, and W. S. Noble. The spectrum kernel: A string kernel for SVM protein classification. *Proceedings of the Pacific Biocomputing Symposium*, 2002.

[15] A. G. Murzin, S. E. Brenner, T. Hubbard, and C. Chothia. SCOP: A structural classification of proteins database for the investigation of sequences and structures. *Journal of Molecular Biology*, 247:536–540, 1995.

[16] M. Sagot. Spelling approximate or repeated motifs using a suffix tree. *Lecture Notes in Computer Science*, 1380:111–127, 1998.

[17] G. Pavesi, G. Mauri, and G. Pesole. An algorithm for finding signals of unknown length in DNA sequences. *Bioinformatics*, 17:S207–S214, July 2001. Proceedings of the Ninth International Conference on Intelligent Systems for Molecular Biology.

[18] S. Henikoff and J. G. Henikoff. Embedding strategies for effective use of information from multiple sequence alignments. *Protein Science*, 6(3):698–705, 1997.

[19] S. L. Salzberg. On comparing classifiers: Pitfalls to avoid and a recommended approach. *Data Mining and Knowledge Discovery*, 1:371–328, 1997.

[20] C. Watkins. Dynamic alignment kernels. Technical report, UL Royal Holloway, 1999.

[21] D. Haussler. Convolution kernels on discrete structure. Technical report, UC Santa Cruz, 1999.

[22] Huma Lodhi, John Shawe-Taylor, Nello Cristianini, and Chris Watkins. Text classification using string kernels. Preprint.
